# Noisy Generalized Binary Search

**Robert Nowak**
University of Wisconsin-Madison
1415 Engineering Drive, Madison WI 53706
nowak@ece.wisc.edu

## Abstract

This paper addresses the problem of noisy Generalized Binary Search (GBS). GBS is a well-known greedy algorithm for determining a binary-valued hypothesis through a sequence of strategically selected queries. At each step, a query is selected that most evenly splits the hypotheses under consideration into two disjoint subsets, a natural generalization of the idea underlying classic binary search. GBS is used in many applications, including fault testing, machine diagnostics, disease diagnosis, job scheduling, image processing, computer vision, and active learning. In most of these cases, the responses to queries can be noisy. Past work has provided a partial characterization of GBS, but existing noise-tolerant versions of GBS are suboptimal in terms of query complexity. This paper presents an optimal algorithm for noisy GBS and demonstrates its application to learning multidimensional threshold functions.

## 1 Introduction

This paper studies learning problems of the following form. Consider a finite, but potentially very large, collection of binary-valued functions $\mathcal{H}$ defined on a domain $\mathcal{X}$. In this paper, $\mathcal{H}$ will be called the *hypothesis space* and $\mathcal{X}$ will be called the *query space*. Each $h \in \mathcal{H}$ is a mapping from $\mathcal{X}$ to $\{-1, 1\}$. Assume that the functions in $\mathcal{H}$ are unique and that one function, $h^* \in \mathcal{H}$, produces the correct binary labeling. The goal is to determine $h^*$ through as few queries from $\mathcal{X}$ as possible. For each query $x \in \mathcal{X}$, the value $h^*(x)$, corrupted with independently distributed binary noise, is observed. If the queries were noiseless, then they are usually called *membership queries* to distinguish them from other types of queries [Ang01]; here we will simply refer to them as queries. Problems of this nature arise in many applications , including channel coding [Hor63], experimental design [Rén61], disease diagnosis [Lov85], fault-tolerant computing [FRPU94], job scheduling [KPB99], image processing [KK00], computer vision [SS93, GJ96], computational geometry [AMM$^+$98], and active learning [Das04, BBZ07, Now08].

Past work has provided a partial characterization of this problem. If the responses to queries are noiseless, then selecting the optimal sequence of queries from $\mathcal{X}$ is equivalent to determining an optimal binary decision tree, where a sequence of queries defines a path from the root of the tree (corresponding to $\mathcal{H}$) to a leaf (corresponding to a single element of $\mathcal{H}$). In general the determination of the optimal tree is NP-complete [HR76]. However, there exists a greedy procedure that yields query sequences that are within an $O(\log |\mathcal{H}|)$ factor of the optimal search tree depth [GG74, KPB99, Lov85, AMM$^+$98, Das04], where $|\mathcal{H}|$ denotes the cardinality of $\mathcal{H}$. The greedy procedure is referred to as *Generalized Binary Search* (GBS) [Das04, Now08] or the *splitting algorithm* [KPB99, Lov85, GG74]), and it reduces to classic binary search in special cases [Now08]. The GBS algorithm is outlined in Figure 1(a). At each step GBS selects a query that results in the most even split of the hypotheses under consideration into two subsets responding $+1$ and $-1$, respectively, to the query. The correct response to the query eliminates one of these two subsets from further consideration. Since the hypotheses are assumed to be distinct, it is clear that GBS terminates in at most $|\mathcal{H}|$ queries (since it is always possible to find query that eliminates at least

| **Generalized Binary Search (GBS)** | **Noisy Generalized Binary Search (NGBS)** |
|---|---|
| initialize: $i = 0, \mathcal{H}_0 = \mathcal{H}$.<br>while $|\mathcal{H}_i| > 1$<br>1) Select $x_i = \arg\min_{x \in \mathcal{X}} | \sum_{h \in \mathcal{H}_i} h(x)|$.<br>2) Obtain response $y_i = h^*(x_i)$.<br>3) Set $\mathcal{H}_{i+1} = \{h \in \mathcal{H}_i : h(x_i) = y_i\}$,<br>$i = i + 1$. | initialize: $p_0$ uniform over $\mathcal{H}$.<br>for $i = 0, 1, 2, \ldots$<br>1) $x_i = \arg\min_{x \in \mathcal{X}} | \sum_{h \in \mathcal{H}} p_i(h) h(x)|$.<br>2) Obtain noisy response $y_i$.<br>3) Bayes update $p_i \to p_{i+1}$; Eqn. (1).<br><br>hypothesis selected at each step:<br>$\widehat{h}_i := \arg\max_{h \in H} p_i(h)$ |
| (a) | (b) |

Figure 1: Generalized binary search (GBS) algorithm and a noise-tolerant variant (NGBS).

one hypothesis at each step). In fact, there are simple examples demonstrating that this is the best one can hope to do in general [KPB99, Lov85, GG74, Das04, Now08]. However, it is also true that in many cases the performance of GBS can be much better [AMM$^+$98, Now08]. In general, the number of queries required can be bounded in terms of a combinatorial parameter of $\mathcal{H}$ called the extended teaching dimension [Ang01, Heg95] (also see [HPRW96] for related work). Alternatively, there exists a geometric relation between the pair $(\mathcal{X}, \mathcal{H})$, called the *neighborly* condition, that is sufficient to bound the number of queries needed [Now08].

The focus of this paper is noisy GBS. In many (if not most) applications it is unrealistic to assume that the responses to queries are without error. Noise-tolerant versions of classic binary search have been well-studied. The classic binary search problem is equivalent to learning a one-dimensional binary-valued threshold function by selecting point evaluations of the function according to a bisection procedure. A noisy version of classic binary search was studied first in the context of channel coding with feedback [Hor63]. Horstein's probabilistic bisection procedure [Hor63] was shown to be optimal (optimal decay of the error probability) [BZ74] (also see[KK07]).

One straightforward approach to noisy GBS was explored in [Now08]. The idea is to follow the GBS algorithm, but to repeat the query at each step multiple times in order to decide whether the response is more probably $+1$ or $-1$. The strategy of repeating queries has been suggested as a general approach for devising noise-tolerant learning algorithms [Kää06]. This simple approach has been studied in the context of noisy versions of classic binary search and shown to be suboptimal [KK07]. Since classic binary search is a special case of the general problem, it follows immediately that the approach proposed in [Now08] is suboptimal. This paper addresses the open problem of determining an optimal strategy for noisy GBS. An optimal noise-tolerant version of GBS is developed here. The number of queries an algorithm requires to confidently identify $h^*$ is called the *query complexity* of the algorithm. The query complexity of the new algorithm is optimal, and we are not aware of any other algorithm with this capability.

It is also shown that optimal convergence rate and query complexity is achieved for a broad class of geometrical hypotheses arising in image recovery and binary classification. Edges in images and decision boundaries in classification problems are naturally viewed as curves in the plane or surfaces embedded in higher-dimensional spaces and can be associated with multidimensional threshold functions valued $+1$ and $-1$ on either side of the curve/surface. Thus, one important setting for GBS is when $\mathcal{X}$ is a subset of $d$ dimensional Euclidean space and the set $\mathcal{H}$ consists of multidimensional threshold functions. We show that our algorithm achieves the optimal query complexity for actively learning multidimensional threshold functions in noisy conditions.

The paper is organized as follows. Section 2 describes the Bayesian algorithm for noisy GBS and presents the main results. Section 3 examines the proposed method for learning multidimensional threshold functions. Section 4 discusses an agnostic algorithm that performs well even if $h^*$ is not in the hypothesis space $\mathcal{H}$. Proofs are given in Section 5.

## 2 A Bayesian Algorithm for Noisy GBS

In noisy GBS, one must cope with erroneous responses. Specifically, assume that the binary response $y \in \{-1, 1\}$ to each query $x \in \mathcal{X}$ is an independent realization of the random variable $Y$ satisfying $\mathbb{P}(Y = h^*(x)) > \mathbb{P}(Y = -h^*(x))$, where $h^* \in \mathcal{H}$ is fixed but unknown. In other words, the response is only probably correct. If a query $x$ is repeated more than once, then each response is

an independent realization of $Y$. Define the *noise-level* for the query $x$ as $\alpha_x := \mathbb{P}(Y = -h^*(x))$. Throughout the paper we will let $\alpha := \sup_{x \in \mathcal{X}} \alpha_x$ and assume that $\alpha < 1/2$.

A Bayesian approach to noisy GBS is investigated in this paper. Let $p_0$ be a known probability measure over $\mathcal{H}$. That is, $p_0 : \mathcal{H} \rightarrow [0, 1]$ and $\sum_{h \in \mathcal{H}} p_0(h) = 1$. The measure $p_0$ can be viewed as an initial weighting over the hypothesis class, expressing the fact that all hypothesis are equally reasonable prior to making queries. After each query and response $(x_i, y_i)$, $i = 0, 1, \ldots$, the distribution is updated according to

$$p_{i+1}(h) \quad \propto \quad p_i(h)\, \beta^{(1-z_i(h))/2}(1 - \beta)^{(1+z_i(h))/2}, \tag{1}$$

where $z_i(h) = h(x_i)y_i$, $h \in \mathcal{H}$, $\beta$ is any constant satisfying $0 < \beta < 1/2$, and $p_{i+1}(h)$ is normalized to satisfy $\sum_{h \in \mathcal{H}} p_{i+1}(h) = 1$. The update can be viewed as an application of Bayes rule and its effect is simple; the probability masses of hypotheses that agree with the label $y_i$ are boosted relative to those that disagree. The parameter $\beta$ controls the size of the boost. The hypothesis with the largest weight is selected at each step: $\widehat{h}_i := \arg\max_{h \in \mathcal{H}} p_i(h)$. If the maximizer is not unique, one of the maximizers is selected at random. The goal of noisy GBS is to drive the error $\mathbb{P}(\widehat{h}_i \neq h^*)$ to zero as quickly as possible by strategically selecting the queries. A similar procedure has been shown to be optimal for noisy (classic) binary search problem [BZ74, KK07]. The crucial distinction here is that GBS calls for a fundamentally different approach to query selection.

The query selection at each step must be informative with respect to the distribution $p_i$. For example, if the *weighted prediction* $\sum_{h \in H} p_i(h)h(x)$ is close to zero for a certain $x$, then a label at that point is informative due to the large disagreement among the hypotheses. This suggests the following noise-tolerant variant of GBS outlined in Figure 1. This paper shows that a slight variation of the query selection in the NGBS algorithm in Figure 1 yields an algorithm with optimal query complexity.

It is shown that as long as $\beta$ is larger than the noise-level of each query, then the NGBS produces a sequence of hypotheses, $\widehat{h}_0, \widehat{h}_1, \ldots$, such that $\mathbb{P}(\widehat{h}_n \neq h^*)$ is bounded above by a monotonically decreasing sequence (see Theorem 1). The main interest of this paper is an algorithm that drives the error to zero exponentially fast, and this requires the query selection criterion to be modified slightly. To see why this is necessary, suppose that at some step of the NGBS algorithm a single hypothesis (e.g., $h^*$) has the majority of the probability mass. Then the weighted prediction will be almost equal to the prediction of that hypothesis (i.e., close to $+1$ or $-1$ for all queries), and therefore the responses to all queries are relatively certain and non-informative. Thus, the convergence of the algorithm could become quite slow in such conditions. A similar effect is true in the case of noisy (classic) binary search [BZ74, KK07]. To address this issue, the query selection criterion is modified via randomization so that the response to the selected query is always highly uncertain.

In order to state the modified selection procedure and the main results, observe that the query space $\mathcal{X}$ can be partitioned into equivalence subsets such that every $h \in \mathcal{H}$ is constant for all queries in each such subset. Let $\mathcal{A}$ denote the smallest such partition. Note that $\mathcal{X} = \bigcup_{A \in \mathcal{A}} A$. For every $A \in \mathcal{A}$ and $h \in \mathcal{H}$, the value of $h(x)$ is constant (either $+1$ or $-1$) for all $x \in A$; denote this value by $h(A)$. As first noted in [Now08], $\mathcal{A}$ can play an important role in GBS. In particular, observe that the query selection step in NGBS is equivalent to an optimization over $\mathcal{A}$ rather that $\mathcal{X}$ itself. The randomization of the query selection step is based on the notion of neighboring sets in $\mathcal{A}$.

**Definition 1** *Two sets $A, A' \in \mathcal{A}$ are said to be* neighbors *if only a single hypothesis (and its complement, if it also belongs to $\mathcal{H}$) outputs a different value on $A$ and $A'$.*

The modified NGBS algorithm is outlined in Figure 2. Note that the query selection step is identical to that of the original NGBS algorithm, unless there exist two neighboring sets with strongly bipolar weighted responses. In the latter case, a query is randomly selected from one of these two sets with equal probability, which guarantees a highly uncertain response.

**Theorem 1** *Let $\mathbb{P}$ denotes the underlying probability measure (governing noises and algorithm randomization). If $\beta > \alpha$, then both the NGBS and modified NGBS algorithms, in Figure 1(b) and Figure 2, respectively, generate a sequence of hypotheses such that $\mathbb{P}(\widehat{h}_n \neq h^*) \leq a_n < 1$, where $\{a_n\}_{n \geq 0}$ is a monotonically decreasing sequence.*

The condition $\beta > \alpha$ ensures that the update (1) is not overly aggressive. We now turn to the matter of sufficient conditions guaranteeing that $\mathbb{P}(\widehat{h}_n \neq h^*) \rightarrow 0$ exponentially fast with $n$. The

---
**Modified NGBS**

initialize: $p_0$ uniform over $\mathcal{H}$.
for $i = 0, 1, 2, \dots$

1) Let $b = \min_{A \in \mathcal{A}} |\sum_{h \in \mathcal{H}} p_i(h) h(A)|$. If there exists neighboring sets $A$ and $A'$ with $\sum_{h \in \mathcal{H}} p_i(h) h(A) > b$ and $\sum_{h \in \mathcal{H}} p_i(h) h(A') < -b$, then select $x_i$ from $A$ or $A'$ with probability $1/2$ each. Otherwise select $x_i$ from the set $A_{\min} = \arg\min_{A \in \mathcal{A}} |\sum_{h \in \mathcal{H}} p_i(h) h(A)|$. In the case that the sets above are non-unique, choose at random any one satisfying the requirements.

2) Obtain noisy response $y_i$.

3) Bayes update $p_i \to p_{i+1}$; Eqn. (1).

---

hypothesis selected at each step:
$\widehat{h}_i := \arg\max_{h \in H} p_i(h)$

---

Figure 2: Modified NGBS algorithm.

exponential convergence rate of classic binary search hinges on the fact that the hypotheses can be ordered with respect to $\mathcal{X}$. In general situations, the hypothesis space cannot be ordered in such a fashion, but the neighborhood graph of $\mathcal{A}$ provides a similar local structure.

**Definition 2** *The pair $(\mathcal{X}, \mathcal{H})$ is said to be* neighborly *if the neighborhood graph of $\mathcal{A}$ is connected (i.e., for every pair of sets in $\mathcal{A}$ there exists a sequence of neighboring sets that begins at one of the pair and ends with the other).*

In essence, the neighborly condition simply means that each hypothesis is locally distinguishable from all others. By 'local' we mean in the vicinity of points $x$ where the output of the hypothesis changes from $+1$ to $-1$. The neighborly condition was first introduced in [Now08] in the analysis of GBS. It is shown in Section 3 that the neighborly condition holds for the important case of hypothesis spaces consisting of multidimensional threshold functions. If $(\mathcal{X}, \mathcal{H})$ is neighborly, then the modified NGBS algorithm guarantees that $\mathbb{P}(\widehat{h}_i \neq h^*) \to 0$ exponentially fast.

**Theorem 2** *Let $\mathbb{P}$ denotes the underlying probability measure (governing noises and algorithm randomization). If $\beta > \alpha$ and $(\mathcal{X}, \mathcal{H})$ is neighborly, then the modified NGBS algorithm in Figure 2 generates a sequence of hypotheses satisfying*

$$\mathbb{P}(\widehat{h}_n \neq h^*) \ \leq \ |\mathcal{H}| \, (1 - \lambda)^n \ \leq \ |\mathcal{H}| \, e^{-\lambda n} \ , \ n = 0, 1, \dots$$

*with exponential constant $\lambda = \min\left\{\frac{1 - c^*}{2}, \frac{1}{4}\right\} \left(1 - \frac{\beta(1 - \alpha)}{1 - \beta} - \frac{\alpha(1 - \beta)}{\beta}\right)$, where*

$$c^* \ := \ \min_P \max_{h \in \mathcal{H}} \left| \int_{\mathcal{X}} h(x) \, dP(x) \right| . \tag{2}$$

The exponential convergence rate[1] is governed by the key parameter $0 \leq c^* < 1$. The minimizer in (2) exists because the minimization can be computed over the space of finite-dimensional probability mass functions over the elements of $\mathcal{A}$. As long as no hypothesis is constant over the whole of $\mathcal{X}$, the value of $c^*$ is typically a small constant much less than 1 that is independent of the size of $\mathcal{H}$ (see [Now08, Now09] and the next section for concrete examples). In such situations, the convergence rate of modified NGBS is optimal, up to constant factors. No other algorithm can solve the noisy GBS problem with a lower query complexity. The query complexity of the modified NGBS algorithm can be derived as follows. Let $\delta > 0$ be a prespecified confidence parameter. The number of queries required to ensure that $\mathbb{P}(\widehat{h}_n \neq h^*) \leq \delta$ is $n \geq \lambda^{-1} \log \frac{|\mathcal{H}|}{\delta} = O(\log \frac{|\mathcal{H}|}{\delta})$, which is the optimal query complexity. Intuitively, $O(\log |\mathcal{H}|)$ bits are required to encode each hypothesis. More formally, the classic noisy binary search problem satisfies the assumptions of Theorem 2 [Now08],

and hence it is a special case of the general problem. It is known that the optimal query complexity for noisy classic binary search is $O(\log \frac{|\mathcal{H}|}{\delta})$ [BZ74, KK07].

We contrast this with the simple noise-tolerant GBS algorithm based on repeating each query in the standard GBS algorithm of Figure 1(a) multiple times to control the noise (see [Kää06, Now08] for related derivations). It follows from Chernoff's bound that the query complexity of determining the correct label for a single query with confidence at least $1 - \delta$ is $O(\frac{\log(1/\delta)}{|1/2-\alpha|^2})$. Suppose that GBS requires $n_0$ queries in the noiseless situation. Then using the union bound, we require $O(\frac{\log(n_0/\delta)}{|1/2-\alpha|^2})$ queries at each step to guarantee that the labels determined for all $n_0$ queries are correct with probability $1 - \delta$. If $(\mathcal{X}, \mathcal{H})$ is neighborly, then GBS requires $n_0 = O(\log |\mathcal{H}|)$ queries in noiseless conditions [Now08]. Therefore, under the conditions of Theorem 2, the query complexity of the simple noise-tolerant GBS algorithm is $O(\log |\mathcal{H}| \log \frac{\log |\mathcal{H}|}{\delta})$, a logarithmic factor worse than the optimal query complexity.

## 3 Noisy GBS for Learning Multidimensional Thresholds

We now apply the theory and modified NGBS algorithm to the problem of learning multidimensional threshold functions from point evaluations, a problem that arises commonly in computer vision [SS93, GJ96, AMM$^+$98], image processing [KK00], and active learning [Das04, BBZ07, CN08, Now08]. In this case, the hypotheses are determined by (possibly nonlinear) decision surfaces in $d$-dimensional Euclidean space (i.e., $\mathcal{X}$ is a subset of $\mathbb{R}^d$), and the queries are points in $\mathbb{R}^d$. It suffices to consider linear decision surfaces of the form $h_{a,b}(x) := \text{sign}(\langle a, x \rangle + b)$, where $a \in \mathbb{R}^d$, $\|a\|_2 = 1$, $b \in \mathbb{R}$, $|b| \le c$ for some constant $c < \infty$, and $\langle a, x \rangle$ denotes the inner product in $\mathbb{R}^d$. Note that hypotheses of this form can be used to represent nonlinear decision surfaces by applying a nonlinear mapping to the query space.

**Theorem 3** *Let $\mathcal{H}$ be a finite collection of hypotheses of form $\text{sign}(\langle a, x \rangle + b)$, for some constant $c < \infty$. Then the hypotheses selected by the modified NGBS algorithm with $\beta > \alpha$ satisfy*

$$\mathbb{P}(\widehat{h}_n \neq h^*) \ \le \ |\mathcal{H}| \, e^{-\lambda n} \,,$$

*with $\lambda = \frac{1}{4} \left( 1 - \frac{\beta(1-\alpha)}{1-\beta} - \frac{\alpha(1-\beta)}{\beta} \right)$. Moreover, $\widehat{h}_n$ can be computed in time polynomial in $|H|$.*

Based on the discussion at the end of the previous section, we conclude that the query complexity of the modified NGBS algorithm is $O(\log |\mathcal{H}|)$; this is the optimal up to constant factors. The only other algorithm with this capability that we are aware of was analyzed in [BBZ07], and it is based on a quite different approach tailored specifically to linear threshold problem.

## 4 Agnostic Algorithms

We also mention the possibility of agnostic algorithms guaranteed to find the best hypothesis in $\mathcal{H}$ even if the optimal hypothesis $h^*$ is not in $\mathcal{H}$ and/or the assumptions of Theorem 2 or 3 do not hold. The best hypothesis in $\mathcal{H}$ is the one that minimizes the error with respect to a given probability measure on $\mathcal{X}$, denoted by $P_X$. The following theorem, proved in [Now09], demonstrates an agnostic algorithm that performs almost as well as empirical risk minimization (ERM) in general, and has the optimal $O(\log |\mathcal{H}|/\delta)$ query complexity when the conditions of Theorem 2 hold.

**Theorem 4** *Let $P_X$ denote a probability distribution on $\mathcal{X}$ and suppose we have a query budget of $n$. Let $h_1$ denote the hypothesis selected by modified NGBS using $n/3$ of the queries and let $h_2$ denote the hypothesis selected by ERM from $n/3$ queries drawn independently from $P_X$. Draw the remaining $n/3$ queries independently from $P_\Delta$, the restriction of $P_X$ to the set $\Delta \subset \mathcal{X}$ on which $h_1$ and $h_2$ disagree, and let $\widehat{R}_\Delta(h_1)$ and $\widehat{R}_\Delta(h_2)$ denote the average number of errors made by $h_1$ and $h_2$ on these queries. Select $\widehat{h} = \arg\min\{\widehat{R}_\Delta(h_1), \widehat{R}_\Delta(h_2)\}$. Then, in general,*

$$\mathbb{E}[R(\widehat{h})] \ \le \ \min\{\mathbb{E}[R(h_1)], \mathbb{E}[R(h_2)]\} + \sqrt{3/n} \,,$$

*where $R(h)$, $h \in \mathcal{H}$, denotes the probability of error of $h$ with respect to $P_X$ and $\mathbb{E}$ denotes the expectation with respect to all random quantities. Furthermore, if the assumptions of Theorem 2 hold with noise bound $\alpha$, then*

$$\mathbb{P}(\widehat{h} \neq h^*) \ \le \ N e^{-\lambda n/3} + 2 e^{-n|1-2\alpha|^2/6} \,.$$

# 5 Appendix: Proofs

## 5.1 Proof of Theorem 1

Let $\mathbb{E}$ denote expectation with respect to $\mathbb{P}$, and define $C_n := (1 - p_n(h^*))/p_n(h^*)$. Note that $C_n \in [0, \infty)$ reflects the amount of mass that $p_n$ places on the suboptimal hypotheses. First note that

$$\mathbb{P}(\widehat{h}_n \neq h^*) \leq \mathbb{P}(p_n(h^*) < 1/2) = \mathbb{P}(C_n > 1) \leq \mathbb{E}[C_n] , \text{ by Markov's inequality.}$$

Next, observe that

$$\begin{aligned}
\mathbb{E}[C_n] &= \mathbb{E}[(C_n/C_{n-1})\,C_{n-1}] = \mathbb{E}\left[\mathbb{E}[(C_n/C_{n-1})\,C_{n-1}|p_{n-1}]\right] \\
&= \mathbb{E}\left[C_{n-1}\,\mathbb{E}[(C_n/C_{n-1})|p_{n-1}]\right] \leq \mathbb{E}[C_{n-1}]\max_{p_{n-1}}\mathbb{E}[(C_n/C_{n-1})|p_{n-1}] \\
&\leq C_0 \left(\max_{i=0,\ldots,n-1}\max_{p_i}\mathbb{E}[(C_{i+1}/C_i)|p_i]\right)^n .
\end{aligned}$$

Note that because $p_0$ is assumed to be uniform, $C_0 = |H| - 1$. A similar conditioning technique is employed for interval estimation in [BZ74]. The rest of the proof entails showing that $\mathbb{E}[(C_{i+1}/C_i)|p_i] < 1$, which proofs the result, and requires a very different approach than [BZ74].

The precise form of $p_1, p_2, \ldots$ is derived as follows. Let $\delta_i = (1 + \sum_h p_i(h)\,z_i(h))/2$, the weighted proportion of hypotheses that agree with $y_i$. The factor that normalizes the updated distribution in (1) is related to $\delta_i$ as follows. Note that $\sum_h p_i(h)\,\beta^{(1-z_i(h))/2}(1-\beta)^{(1+z_i(h))/2} = \sum_{h:z_i(h)=-1} p_i(h)\beta + \sum_{h:z_i(h)=1} p_i(h)(1-\beta) = (1-\delta_i)\beta + \delta_i(1-\beta)$. Thus,

$$p_{i+1}(h) = p_i(h)\,\frac{\beta^{(1-z_i(h))/2}(1-\beta)^{(1+z_i(h))/2}}{(1-\delta_i)\beta + \delta_i(1-\beta)}$$

Denote the reciprocal of the update factor for $p_{i+1}(h^*)$ by

$$\gamma_i := \frac{(1-\delta_i)\beta + \delta_i(1-\beta)}{\beta^{(1-Z_i(h^*))/2}(1-\beta)^{(1+Z_i(h^*))/2}} , \tag{3}$$

where $z_i(h^*) = h^*(x_i)y_i$, and observe that $p_{i+1}(h^*) = p_i(h^*)/\gamma_i$. Thus,

$$\frac{C_{i+1}}{C_i} = \frac{(1 - p_i(h^*)/\gamma_i)p_i(h^*)}{p_i(h^*)/\gamma_i(1 - p_i(h^*))} = \frac{\gamma_i - p_i(h^*)}{1 - p_i(h^*)} .$$

Now to bound $\max_{p_i}\mathbb{E}[C_{i+1}/C_i|p_i] < 1$ we will show that $\max_{p_i}\mathbb{E}[\gamma_i|p_i] < 1$. To accomplish this, we will assume that $p_i$ is arbitrary.

For every $A \in \mathcal{A}$ and every $h \in \mathcal{H}$ let $h(A)$ denote the value of $h$ on the set $A$. Define $\delta_A^+ = (1 + \sum_h p_i(h)h(A))/2$, the proportion of hypotheses that take the value $+1$ on $A$. Note that for every $A$ we have $0 < \delta_A^+ < 1$, since at least one hypothesis has the value $-1$ on $A$ and $p(h) > 0$ for all $h \in \mathcal{H}$. Let $A_i$ denote that set that $x_i$ is selected from, and consider the four possible situations:

$$\begin{aligned}
h^*(x_i) = +1,\ y_i = +1: \quad &\gamma_i = \frac{(1-\delta_{A_i}^+)\beta + \delta_{A_i}^+(1-\beta)}{1-\beta} \\
h^*(x_i) = +1,\ y_i = -1: \quad &\gamma_i = \frac{\delta_{A_i}^+\beta + (1-\delta_{A_i}^+)(1-\beta)}{\beta} \\
h^*(x_i) = -1,\ y_i = +1: \quad &\gamma_i = \frac{(1-\delta_{A_i}^+)\beta + \delta_{A_i}^+(1-\beta)}{\beta} \\
h^*(x_i) = -1,\ y_i = -1: \quad &\gamma_i = \frac{\delta_{A_i}^+\beta + (1-\delta_{A_i}^+)(1-\beta)}{1-\beta}
\end{aligned}$$

To bound $\mathbb{E}[\gamma_i|p_i]$ it is helpful to condition on $A_i$. Define $q_i := \mathbb{P}_{x,y|A_i}(h^*(x) \neq Y)$. If $h^*(A_i) = +1$, then

$$\begin{aligned}
\mathbb{E}[\gamma_i|p_i, A_i] &= \frac{(1-\delta_{A_i}^+)\beta + \delta_{A_i}^+(1-\beta)}{1-\beta}(1-q_i) + \frac{\delta_{A_i}^+\beta + (1-\delta_{A_i}^+)(1-\beta)}{\beta}q_i \\
&= \delta_{A_i}^+ + (1-\delta_{A_i}^+)\left[\frac{\beta(1-q_i)}{1-\beta} + \frac{q_i(1-\beta)}{\beta}\right] .
\end{aligned}$$

Define $\gamma_i^+(A_i) := \delta_{A_i}^+ + (1 - \delta_{A_i}^+)\left[\frac{\beta(1-q_i)}{1-\beta} + \frac{q_i(1-\beta)}{\beta}\right]$. Similarly, if $h^*(A_i) = -1$, then

$$\mathbb{E}[\gamma_i|p_i, A_i] \;=\; (1 - \delta_{A_i}^+) + \delta_{A_i}^+ \left[\frac{\beta(1-q_i)}{1-\beta} + \frac{q_i(1-\beta)}{\beta}\right] \;=:\; \gamma_i^-(A_i)$$

By assumption $q_i \leq \alpha < 1/2$, and since $\alpha < \beta < 1/2$ the factor $\frac{\beta(1-q_i)}{1-\beta} + \frac{q_i(1-\beta)}{\beta} \leq \frac{\beta(1-\alpha)}{1-\beta} + \frac{\alpha(1-\beta)}{\beta} < 1$. Define

$$\varepsilon_0 := 1 - \frac{\beta(1-\alpha)}{1-\beta} - \frac{\alpha(1-\beta)}{\beta}\,,$$

to obtain the bounds

$$\gamma_i^+(A_i) \;\leq\; \delta_{A_i}^+ + (1 - \delta_{A_i}^+)(1 - \varepsilon_0)\,, \tag{4}$$
$$\gamma_i^-(A_i) \;\leq\; \delta_{A_i}^+(1 - \varepsilon_0) + (1 - \delta_{A_i}^+)\,. \tag{5}$$

Since both $\gamma_i^+(A_i)$ and $\gamma_i^-(A_i)$ are less than 1, it follows that $\mathbb{E}[\gamma_i|p_i] < 1$.  □

## 5.2  Proof of Theorem 2

The proof amounts to obtaining upper bounds for $\gamma_i^+(A_i)$ and $\gamma_i^-(A_i)$, defined above in (4) and (5). For every $A \in \mathcal{A}$ and any probability measure $p$ on $\mathcal{H}$ the *weighted prediction* on $A$ is defined to be $W(p, A) := \sum_{h \in H} p(h)h(A)$, where $h(A)$ is the constant value of $h$ for every $x \in A$. The following lemma plays a crucial role in the analysis of the modified NGBS algorithm.

**Lemma 1**  *If $(\mathcal{X}, \mathcal{H})$ is neighborly, then for every probability measure $p$ on $\mathcal{H}$ there either exists a set $A \in \mathcal{A}$ such that $|W(p, A)| \leq c^*$ or a pair of neighboring sets $A, A' \in \mathcal{A}$ such that $W(p, A) > c^*$ and $W(p, A') < -c^*$.*

**Proof of Lemma 1**: Suppose that $\min_{A \in \mathcal{A}} |W(p, A)| > c^*$. Then there must exist $A, A' \in \mathcal{A}$ such that $W(p, A) > c^*$ and $W(p, A') < -c^*$, otherwise $c^*$ cannot be the minimax moment of $\mathcal{H}$. To see this suppose, for instance, that $W(p, A) > c^*$ for all $A \in \mathcal{A}$. Then for every distribution $P$ on $\mathcal{X}$ we have $\int_{\mathcal{X}} \sum_{h \in \mathcal{H}} p(h)h(x)dP(x) > c^*$. This contradicts the definition of $c^*$ since $\int_{\mathcal{X}} \sum_{h \in \mathcal{H}} p(h)h(x)dP(x) \leq \sum_{h \in \mathcal{H}} p(h)|\int_{\mathcal{X}} h(x)\,dP(x)| \leq \max_{h \in \mathcal{H}} |\int_{\mathcal{X}} h(x)\,dP(x)|$. The neighborly condition guarantees that there exists a sequence of neighboring sets beginning at $A$ and ending at $A'$. Since $|W(p, A)| > c^*$ on every set and the sign of $W(p, \cdot)$ must change at some point in the sequence, it follows that there exist neighboring sets satisfying the claim.  □

Now consider two distinct situations. Define $b_i := \min_{A \in \mathcal{A}} |W(p_i, A)|$. First suppose that there do not exist neighboring sets $A$ and $A'$ with $W(p_i, A) > b_i$ and $W(p_i, A') < -b_i$. Then by Lemma 1, this implies that $b_i \leq c^*$, and according the query selection step of the modified NGBS algorithm, $A_i = \arg \min_A |W(p_i, A)|$. Note that because $|W(p_i, A_i)| \leq c^*$, $(1 - c^*)/2 \leq \delta_{A_i}^+ \leq (1 + c^*)/2$. Hence, both $\gamma_i^+(A_i)$ and $\gamma_i^-(A_i)$ are bounded above by $1 - \varepsilon_0(1 - c^*)/2$.

Now suppose that there exist neighboring sets $A$ and $A'$ with $W(p_i, A) > b_i$ and $W(p_i, A') < -b_i$. Recall that in this case $A_i$ is randomly chosen to be $A$ or $A'$ with equal probability. Note that $\delta_A^+ > (1 + b_i)/2$ and $\delta_{A'}^+ < (1 - b_i)/2$. If $h^*(A) = h^*(A') = +1$, then applying (4) results in

$$\mathbb{E}[\gamma_i|p_i, A_i \in \{A, A'\}] \;<\; \frac{1}{2}\left(1 + \frac{1 - b_i}{2} + \frac{1 + b_i}{2}(1 - \varepsilon_0)\right) \;=\; \frac{1}{2}\left(2 - \varepsilon_0 \frac{1 + b_i}{2}\right) \;\leq\; 1 - \varepsilon_0/4\,,$$

since $b_i > 0$. Similarly, if $h^*(A) = h^*(A') = -1$, then (5) yields $\mathbb{E}[\gamma_i|p_i, A_i \in \{A, A'\}] < 1 - \varepsilon_0/4$. If $h^*(A) = -1$ on $A$ and $h^*(A') = +1$, then applying (5) on $A$ and (4) on $A'$ yields

$$
\begin{aligned}
\mathbb{E}[\gamma_i|p_i, A_i \in \{A, A'\}] \;&\leq\; \frac{1}{2}\left(\delta_A^+(1 - \varepsilon_0) + (1 - \delta_A^+) + \delta_{A'}^+ + (1 - \delta_{A'}^+)(1 - \varepsilon_0)\right) \\
&=\; \frac{1}{2}\left(1 - \delta_A^+ + \delta_{A'}^+ + (1 - \varepsilon_0)(1 + \delta_A^+ - \delta_{A'}^+)\right) \\
&=\; \frac{1}{2}\left(2 - \varepsilon_0(1 + \delta_A^+ - \delta_{A'}^+)\right) \\
&=\; 1 - \frac{\varepsilon_0}{2}(1 + \delta_A^+ - \delta_{A'}^+) \;\leq\; 1 - \varepsilon_0/2\,,
\end{aligned}
$$

since $0 \leq \delta_A^+ - \delta_{A'}^+ \leq 1$. The final possibility is that $h^*(A) = +1$ and $h^*(A') = -1$. Apply (4) on $A$ and (5) on $A'$ to obtain

$$
\begin{aligned}
\mathbb{E}[\gamma_i | p_i, A_i \in \{A, A'\}] &\leq \frac{1}{2}\left(\delta_A^+ + (1 - \delta_A^+)(1 - \varepsilon_0) + \delta_{A'}^+(1 - \varepsilon_0) + (1 - \delta_{A'}^+)\right) \\
&= \frac{1}{2}(1 + \delta_A^+ - \delta_{A'}^+ + (1 - \varepsilon_0)(1 - \delta_A^+ + \delta_{A'}^+))
\end{aligned}
$$

Next, use the fact that because $A$ and $A'$ are neighbors, $\delta_A^+ - \delta_{A'}^+ = p_i(h^*) - p_i(-h^*)$; if $-h^*$ does not belong to $\mathcal{H}$, then $p_i(-h^*) = 0$. Hence,

$$
\begin{aligned}
\mathbb{E}[\gamma_i | p_i, A_i \in \{A, A'\}] &\leq \frac{1}{2}(1 + \delta_A^+ - \delta_{A'}^+ + (1 - \epsilon_0)(1 - \delta_A^+ + \delta_{A'}^+)) \\
&= \frac{1}{2}(1 + p_i(h^*) - p_i(-h^*) + (1 - \epsilon_0)(1 - p_i(h^*) + p_i(-h^*))) \\
&\leq \frac{1}{2}(1 + p_i(h^*) + (1 - \epsilon_0)(1 - p_i(h^*))) = 1 - \frac{\varepsilon_0}{2}(1 - p_i(h^*)),
\end{aligned}
$$

since the bound is maximized when $p_i(-h^*) = 0$. Now bound $\mathbb{E}[\gamma_i | p_i]$ by the maximum of the conditional bounds above to obtain

$$
\mathbb{E}[\gamma_i | p_i] \leq \max\left\{1 - \frac{\varepsilon_0}{2}(1 - p_i(h^*)), \ 1 - \frac{\varepsilon_0}{4}, \ 1 - (1 - c^*)\frac{\varepsilon_0}{2}\right\},
$$

and thus it is easy to see that

$$
\mathbb{E}\left[\frac{C_{i+1}}{C_i}\middle| p_i\right] = \frac{\mathbb{E}[\gamma_i | p_i] - p_i(h^*)}{1 - p_i(h^*)} \leq 1 - \min\left\{\frac{\varepsilon_0}{2}(1 - c^*), \frac{\varepsilon_0}{4}\right\}. \qquad \square
$$

### 5.3 Proof of Theorem 3

First we show that the pair $(\mathbb{R}^d, \mathcal{H})$ is neighborly (Definition 2). Each $A \in \mathcal{A}$ is a polytope in $\mathbb{R}^d$. These polytopes are generated by intersections of the halfspaces corresponding to the hypotheses. Any two polytopes that share a common face are neighbors (the hypothesis whose decision boundary defines the face, and its complement if it exists, are the only ones that predict different values on these two sets). Since the polytopes tessellate $\mathbb{R}^d$, the neighborhood graph of $\mathcal{A}$ is connected.

Next consider the final bound in the proof of Theorem 2, above. We next show that the value of $c^*$, defined in (2), is 0. Since the offsets $b$ of the hypotheses are all less than $c$ in magnitude, it follows that the distance from the origin to the nearest point of the decision surface of every hypothesis is at most $c$. Let $P_r$ denote the uniform probability distribution on a ball of radius $r$ centered at the origin in $\mathbb{R}^d$. Then for every $h$ of the form $\text{sign}(\langle a, x \rangle + b)$

$$
\left|\int_{\mathbb{R}^d} h(x)\, dP_r(x)\right| \leq \frac{c}{r},
$$

and $\lim_{r \to \infty} \left|\int_{\mathcal{X}} h(x)\, dP_r(x)\right| = 0$ and so $c^* = 0$.

Lastly, note that the modified NGBS algorithm involves computing $\sum_{h \in H} p_i(h)h(A)$ for all $A \in \mathcal{A}$ at each step. The computational complexity of each step is therefore proportional to the cardinality of $\mathcal{A}$, which is equal to the number of polytopes generated by intersections of half-spaces. It is known that $|\mathcal{A}| = \sum_{i=0}^d \binom{|H|}{i} = O(|H|^d)$ [Buc43]. $\qquad \square$

## Footnotes

[1] Note that the factor $\left(1 - \frac{\beta(1 - \alpha)}{1 - \beta} - \frac{\alpha(1 - \beta)}{\beta}\right)$ in the exponential rate parameter $\lambda$ is a positive constant strictly less than 1. For a noise level $\alpha$ this factor is maximized by a value $\beta \in (\alpha, 1/2)$ which tends to $(1/2 + \alpha)/2$ as $\alpha$ tends to $1/2$.

# References

[AMM+98] E. M. Arkin, H. Meijer, J. S. B. Mitchell, D. Rappaport, and S.S. Skiena. Decision trees for geometric models. *Intl. J. Computational Geometry and Applications*, 8(3):343–363, 1998.

[Ang01] D. Angluin. Queries revisited. *Springer Lecture Notes in Comp. Sci.: Algorithmic Learning Theory*, pages 12–31, 2001.

[BBZ07] M.-F. Balcan, A. Broder, and T. Zhang. Margin based active learning. In *Conf. on Learning Theory (COLT)*, 2007.

[Buc43] R. C. Buck. Partition of space. *The American Math. Monthly*, 50(9):541–544, 1943.

[BZ74] M. V. Burnashev and K. Sh. Zigangirov. An interval estimation problem for controlled observations. *Problems in Information Transmission*, 10:223–231, 1974.

[CN08] R. Castro and R. Nowak. Minimax bounds for active learning. *IEEE Trans. Info. Theory*, pages 2339–2353, 2008.

[Das04] S. Dasgupta. Analysis of a greedy active learning strategy. In *Neural Information Processing Systems*, 2004.

[FRPU94] U. Feige, E. Raghavan, D. Peleg, and E. Upfal. Computing with noisy information. *SIAM J. Comput.*, 23(5):1001–1018, 1994.

[GG74] M. R. Garey and R. L. Graham. Performance bounds on the splitting algorithm for binary testing. *Acta Inf.*, 3:347–355, 1974.

[GJ96] D. Geman and B. Jedynak. An active testing model for tracking roads in satellite images. *IEEE Trans. PAMI*, 18(1):1–14, 1996.

[Heg95] T. Hegedüs. Generalized teaching dimensions and the query complexity of learning. In *8th Annual Conference on Computational Learning Theory*, pages 108–117, 1995.

[Hor63] M. Horstein. Sequential decoding using noiseless feedback. *IEEE Trans. Info. Theory*, 9(3):136–143, 1963.

[HPRW96] L. Hellerstein, K. Pillaipakkamnatt, V. Raghavan, and D. Wilkins. How many queries are needed to learn? *J. ACM*, 43(5):840–862, 1996.

[HR76] L. Hyafil and R. L. Rivest. Constructing optimal binary decision trees is NP-complete. *Inf. Process. Lett.*, 5:15–17, 1976.

[Kää06] M. Kääriäinen. Active learning in the non-realizable case. In *Algorithmic Learning Theory*, pages 63–77, 2006.

[KK00] A. P. Korostelev and J.-C. Kim. Rates of convergence fo the sup-norm risk in image models under sequential designs. *Statistics & Probability Letters*, 46:391–399, 2000.

[KK07] R. Karp and R. Kleinberg. Noisy binary search and its applications. In *Proceedings of the 18th ACM-SIAM Symposium on Discrete Algorithms (SODA 2007)*, pages 881–890, 2007.

[KPB99] S. R. Kosaraju, T. M. Przytycka, and R. Borgstrom. On an optimal split tree problem. *Lecture Notes in Computer Science: Algorithms and Data Structures*, 1663:157–168, 1999.

[Lov85] D. W. Loveland. Performance bounds for binary testing with arbitrary weights. *Acta Informatica*, 22:101–114, 1985.

[Now08] R. Nowak. Generalized binary search. In *Proceedings of the 46th Allerton Conference on Communications, Control, and Computing*, pages 568–574, 2008.

[Now09] R. Nowak. The geometry of generalized binary search. 2009. Preprint available at http://arxiv.org/abs/0910.4397.

[Rén61] A. Rényi. On a problem in information theory. *MTA Mat. Kut. Int. Kozl.*, page 505516, 1961. reprinted in *Selected Papers of Alfred Rényi*, vol. 2, P. Turan, ed., pp. 631-638. Akademiai Kiado, Budapest, 1976.

[SS93] M.J. Swain and M.A. Stricker. Promising directions in active vision. *Int. J. Computer Vision*, 11(2):109–126, 1993.

